# Clustered Multi-Task Learning Via Alternating Structure Optimization

**Jiayu Zhou, Jianhui Chen, Jieping Ye**
Computer Science and Engineering
Arizona State University
Tempe, AZ 85287
{jiayu.zhou, jianhui.chen, jieping.ye}@asu.edu

## Abstract

Multi-task learning (MTL) learns multiple related tasks simultaneously to improve generalization performance. Alternating structure optimization (ASO) is a popular MTL method that learns a shared low-dimensional predictive structure on hypothesis spaces from multiple related tasks. It has been applied successfully in many real world applications. As an alternative MTL approach, clustered multi-task learning (CMTL) assumes that multiple tasks follow a clustered structure, i.e., tasks are partitioned into a set of groups where tasks in the same group are similar to each other, and that such a clustered structure is unknown a priori. The objectives in ASO and CMTL differ in how multiple tasks are related. Interestingly, we show in this paper the equivalence relationship between ASO and CMTL, providing significant new insights into ASO and CMTL as well as their inherent relationship. The CMTL formulation is non-convex, and we adopt a convex relaxation to the CMTL formulation. We further establish the equivalence relationship between the proposed convex relaxation of CMTL and an existing convex relaxation of ASO, and show that the proposed convex CMTL formulation is significantly more efficient especially for high-dimensional data. In addition, we present three algorithms for solving the convex CMTL formulation. We report experimental results on benchmark datasets to demonstrate the efficiency of the proposed algorithms.

## 1  Introduction

Many real-world problems involve multiple related classificatrion/regression tasks. A naive approach is to apply single task learning (STL) where each task is solved independently and thus the task relatedness is not exploited. Recently, there is a growing interest in multi-task learning (MTL), where we learn multiple related tasks simultaneously by extracting appropriate shared information across tasks. In MTL, multiple tasks are expected to benefit from each other, resulting in improved generalization performance. The effectiveness of MTL has been demonstrated empirically [1, 2, 3] and theoretically [4, 5, 6]. MTL has been applied in many applications including biomedical informatics [7], marketing [1], natural language processing [2], and computer vision [3].

Many different MTL approaches have been proposed in the past; they differ in how the relatedness among different tasks is modeled. Evgeniou *et al.* [8] proposed the regularized MTL which constrained the models of all tasks to be close to each other. The task relatedness can also be modeled by constraining multiple tasks to share a common underlying structure [4, 6, 9, 10]. Ando and Zhang [5] proposed a *structural learning* formulation, which assumed multiple predictors for different tasks shared a common structure on the underlying predictor space. For linear predictors, they proposed the alternating structure optimization (ASO) that simultaneously performed inference on multiple tasks and discovered the shared low-dimensional predictive structure. ASO has been

shown to be effective in many practical applications [2, 11, 12]. One limitation of the original ASO formulation is that it involves a non-convex optimization problem and a globally optimal solution is not guaranteed. A convex relaxation of ASO called CASO was proposed and analyzed in [13].

Many existing MTL formulations are based on the assumption that all tasks are related. In practical applications, the tasks may exhibit a more sophisticated group structure where the models of tasks from the same group are closer to each other than those from a different group. There have been many prior work along this line of research, known as clustered multi-task learning (CMTL). In [14], the mutual relatedness of tasks was estimated and knowledge of one task could be transferred to other tasks in the same cluster. Bakker and Heskes [15] used clustered multi-task learning in a Bayesian setting by considering a mixture of Gaussians instead of single Gaussian priors. Evgeniou *et al.* [8] proposed the task clustering regularization and showed how cluster information could be encoded in MTL, and however the group structure was required to be known a priori. Xue *et al.* [16] introduced the Dirichlet process prior which automatically identified subgroups of related tasks. In [17], a clustered MTL framework was proposed that simultaneously identified clusters and performed multi-task inference. Because the formulation is non-convex, they also proposed a convex relaxation to obtain a global optimum [17]. Wang *et al.* [18] used a similar idea to consider clustered tasks by introducing an inter-task regularization.

The objective in CMTL differs from many MTL formulations (e.g., ASO which aims to identify a shared low-dimensional predictive structure for all tasks) which are based on the standard assumption that each task can learn equally well from any other task. In this paper, we study the inherent relationship between these two seemingly different MTL formulations. Specifically, we establish the equivalence relationship between ASO and a specific formulation of CMTL, which performs simultaneous multi-task learning and task clustering: First, we show that CMTL performs clustering on the tasks, while ASO performs projection on the features to find a shared low-rank structure. Next, we show that the spectral relaxation of the clustering (on tasks) in CMTL and the projection (on the features) in ASO lead to an identical regularization, related to the negative Ky Fan $k$-norm of the weight matrix involving all task models, thus establishing their equivalence relationship. The presented analysis provides significant new insights into ASO and CMTL as well as their inherent relationship. To our best knowledge, the clustering view of ASO has not been explored before.

One major limitation of the ASO/CMTL formulation is that it involves a non-convex optimization, as the negative Ky Fan $k$-norm is concave. We propose a convex relaxation of CMTL, and establish the equivalence relationship between the proposed convex relaxation of CMTL and the convex ASO formulation proposed in [13]. We show that the proposed convex CMTL formulation is significantly more efficient especially for high-dimensional data. We further develop three algorithms for solving the convex CMTL formulation based on the block coordinate descent, accelerated projected gradient, and gradient descent, respectively. We have conducted experiments on benchmark datasets including School and Sarcos; our results demonstrate the efficiency of the proposed algorithms.

**Notation:** Throughout this paper, $\mathbb{R}^d$ denotes the $d$-dimensional Euclidean space. $I$ denotes the identity matrix of a proper size. $\mathbb{N}$ denotes the set of natural numbers. $\mathbb{S}_+^m$ denotes the set of symmetric positive semi-definite matrices of size $m$ by $m$. $A \preceq B$ means that $B - A$ is positie semi-definite. $\mathrm{tr}\,(X)$ is the trace of $X$.

## 2    Multi-Task Learning: ASO and CMTL

Assume we are given a multi-task learning problem with $m$ tasks; each task $i \in \mathbb{N}_m$ is associated with a set of training data $\{(x_1^i, y_1^i), \ldots, (x_{n_i}^i, y_{n_i}^i)\} \subset \mathbb{R}^d \times \mathbb{R}$, and a linear predictive function $f_i$: $f_i(x_j^i) = w_i^T x_j^i$, where $w_i$ is the weight vector of the $i$-th task, $d$ is the data dimensionality, and $n_i$ is the number of samples of the $i$-th task. We denote $W = [w_1, \ldots, w_m] \in \mathbb{R}^{d \times m}$ as the weight matrix to be estimated. Given a loss function $\ell(\cdot, \cdot)$, the empirical risk is given by:

$$\mathcal{L}(W) = \sum_{i=1}^m \frac{1}{n_i} \left( \sum_{j=1}^{n_i} \ell(w_i^T x_j^i, y_j^i) \right).$$

We study the following multi-task learning formulation: $\min_W \mathcal{L}(W) + \Omega(W)$, where $\Omega$ encodes our prior knowledge about the $m$ tasks. Next, we review ASO and CMTL and explore their inherent relationship.

## 2.1 Alternating structure optimization

In ASO [5], all tasks are assumed to share a common feature space $\Theta \in \mathbb{R}^{h \times d}$, where $h \leq \min(m, d)$ is the dimensionality of the shared feature space and $\Theta$ has orthonormal columns, i.e., $\Theta\Theta^T = I_h$. The predictive function of ASO is: $f_i(x_j^i) = w_i^T x_j^i = u_i^T x_j^i + v_i^T \Theta x_j^i$, where the weight $w_i = u_i + \Theta^T v_i$ consists of two components including the weight $u_i$ for the high-dimensional feature space and the weight $v_i$ for the low-dimensional space based on $\Theta$. ASO minimizes the following objective function: $\mathcal{L}(W) + \alpha \sum_{i=1}^{m} \|u_i\|_2^2$, subject to: $\Theta\Theta^T = I_h$, where $\alpha$ is the regularization parameter for task relatedness. We can further improve the formulation by including a penalty, $\beta \sum_{i=1}^{m} \|w_i\|_2^2$, to improve the generalization performance as in traditional supervised learning. Since $u_i = w_i - \Theta^T v_i$, we obtain the following ASO formulation:

$$\min_{W, \{v_i\}, \Theta : \Theta\Theta^T = I_h} \mathcal{L}(W) + \sum_{i=1}^{m} \left( \alpha \|w_i - \Theta^T v_i\|_2^2 + \beta \|w_i\|_2^2 \right). \tag{1}$$

## 2.2 Clustered multi-task learning

In CMTL, we assume that the tasks are clustered into $k < m$ clusters, and the index set of the $j$-th cluster is defined as $\mathcal{I}_j = \{v | v \in \text{cluster } j\}$. We denote the mean of the $j$th cluster to be $\bar{w}_j = \frac{1}{n_j} \sum_{v \in \mathcal{I}_j} w_v$. For a given $W = [w_1, \cdots, w_m]$, the sum-of-square error (SSE) function in $K$-means clustering is given by [19, 20]:

$$\sum_{j=1}^{k} \sum_{v \in \mathcal{I}_j} \|w_v - \bar{w}_j\|_2^2 = \text{tr}\left(W^T W\right) - \text{tr}\left(F^T W^T W F\right), \tag{2}$$

where the matrix $F \in \mathbb{R}^{m \times k}$ is an orthogonal cluster indicator matrix with $F_{i,j} = \frac{1}{\sqrt{n_j}}$ if $i \in \mathcal{I}_j$ and $F_{i,j} = 0$ otherwise. If we ignore the special structure of $F$ and keep the orthogonality requirement only, the relaxed SSE minimization problem is:

$$\min_{F : F^T F = I_k} \text{tr}\left(W^T W\right) - \text{tr}\left(F^T W^T W F\right), \tag{3}$$

resulting in the following penalty function for CMTL:

$$\Omega_{\text{CMTL}_0}(W, F) = \alpha \left( \text{tr}\left(W^T W\right) - \text{tr}\left(F^T W^T W F\right) \right) + \beta \, \text{tr}\left(W^T W\right), \tag{4}$$

where the first term is derived from the $K$-means clustering objective and the second term is to improve the generalization performance. Combing Eq. (4) with the empirical error term $\mathcal{L}(W)$, we obtain the following CMTL formulation:

$$\min_{W, F : F^T F = I_k} \mathcal{L}(W) + \Omega_{\text{CMTL}_0}(W, F). \tag{5}$$

## 2.3 Equivalence of ASO and CMTL

In the ASO formulation in Eq. (1), it is clear that the optimal $v_i$ is given by $v_i^* = \Theta w_i$. Thus, the penalty in ASO has the following equivalent form:

$$\Omega_{\text{ASO}}(W, \Theta) = \sum_{i=1}^{m} \left( \alpha \|w_i - \Theta^T \Theta w_i\|_2^2 + \beta \|w_i\|_2^2 \right)$$
$$= \alpha \left( \text{tr}\left(W^T W\right) - \text{tr}\left(W^T \Theta^T \Theta W\right) \right) + \beta \, \text{tr}\left(W^T W\right), \tag{6}$$

resulting in the following equivalent ASO formulation:

$$\min_{W, \Theta : \Theta\Theta^T = I_h} \mathcal{L}(W) + \Omega_{\text{ASO}}(W, \Theta). \tag{7}$$

The penalty of the ASO formulation in Eq. (7) looks very similar to the penalty of the CMTL formulation in Eq. (5), however the operations involved are fundamentally different. In the CMTL formulation in Eq. (5), the matrix $F$ is operated on the task dimension, as it is derived from the $K$-means clustering on the tasks; while in the ASO formulation in Eq. (7), the matrix $\Theta$ is operated on the feature dimension, as it aims to identify a shared low-dimensional predictive structure for all tasks. Although different in the mathematical formulation, we show in the following theorem that the objectives of CMTL and ASO are equivalent.

**Theorem 2.1.** *The objectives of CMTL in Eq. (5) and ASO in Eq. (7) are equivalent if the cluster number, $k$, in $K$-means equals to the size, $h$, of the shared low-dimensional feature space.*

*Proof.* Denote $\mathcal{Q}(W) = \mathcal{L}(W) + (\alpha + \beta) \operatorname{tr} \left( W^T W \right)$, with $\alpha, \beta > 0$. Then, CMTL and ASO solve the following optimization problems:

$$\min_{W,F:F^T F=I_p} \mathcal{Q}(W) - \alpha \operatorname{tr} \left( W F F^T W^T \right), \quad \min_{W,\Theta:\Theta\Theta^T=I_p} \mathcal{Q}(W) - \alpha \operatorname{tr} \left( W^T \Theta^T \Theta W \right),$$

respectively. Note that in both CMTL and ASO, the first term $\mathcal{Q}$ is independent of $F$ or $\Theta$, for a given $W$. Thus, the optimal $F$ and $\Theta$ for these two optimization problems are given by solving:

$$[\text{CMTL}] \quad \max_{F:F^T F=I_k} \operatorname{tr} \left( W F F^T W^T \right), \quad [\text{ASO}] \quad \max_{\Theta:\Theta\Theta^T=I_k} \operatorname{tr} \left( W^T \Theta^T \Theta W \right).$$

Since $WW^T$ and $W^T W$ share the same set of nonzero eigenvalues, by the Ky-Fan Theorem [21], both problems above achieve exactly the same maximum objective value: $\|W^T W\|_{(k)} = \sum_{i=1}^{k} \lambda_i(W^T W)$, where $\lambda_i(W^T W)$ denotes the $i$-th largest eigenvalue of $W^T W$ and $\|W^T W\|_{(k)}$ is known as the Ky Fan $k$-norm of matrix $W^T W$. Plugging the results back to the original objective, the optimization problem for both CMTL and ASO becomes $\min_W \mathcal{Q}(W) - \alpha \|W^T W\|_{(k)}$. This completes the proof of this theorem. $\square$

## 3 Convex Relaxation of CMTL

The formulation in Eq. (5) is non-convex. A natural approach is to perform a convex relaxation on CMTL. We first reformulate the penalty in Eq. (5) as follows:

$$\Omega_{\text{CMTL}_0}(W, F) = \alpha \operatorname{tr} \left( W((1 + \eta)I - FF^T)W^T \right), \tag{8}$$

where $\eta$ is defined as $\eta = \beta/\alpha > 0$. Since $F^T F = I_k$, the following holds:

$$(1 + \eta)I - FF^T = \eta(1 + \eta)(\eta I + FF^T)^{-1}.$$

Thus, we can reformulate $\Omega_{\text{CMTL}_0}$ in Eq. (8) as the following equivalent form:

$$\Omega_{\text{CMTL}_1}(W, F) = \alpha\eta(1 + \eta) \operatorname{tr} \left( W(\eta I + FF^T)^{-1}W^T \right). \tag{9}$$

resulting in the following equivalent CMTL formulation:

$$\min_{W,F:F^T F=I_k} \mathcal{L}(W) + \Omega_{\text{CMTL}_1}(W, F). \tag{10}$$

Following [13, 17], we obtain the following convex relaxation of Eq. (10), called cCMTL:

$$\min_{W,M} \mathcal{L}(W) + \Omega_{\text{cCMTL}}(W, M) \ \text{s.t.} \ \operatorname{tr}(M) = k, M \preceq I, M \in \mathbb{S}_+^m. \tag{11}$$

where $\Omega_{\text{cCMTL}}(W, M)$ is defined as:

$$\Omega_{\text{cCMTL}}(W, M) = \alpha\eta(1 + \eta) \operatorname{tr} \left( W(\eta I + M)^{-1}W^T \right). \tag{12}$$

The optimization problem in Eq. (11) is jointly convex with respect to $W$ and $M$ [9].

### 3.1 Equivalence of cASO and cCMTL

A convex relaxation (cASO) of the ASO formulation in Eq. (7) has been proposed in [13]:

$$\min_{W,S} \mathcal{L}(W) + \Omega_{\text{cASO}}(W, S) \ \text{s.t.} \ \operatorname{tr}(S) = h, S \preceq I, \ S \in \mathbb{S}_+^d, \tag{13}$$

where $\Omega_{\text{cASO}}$ is defined as:

$$\Omega_{\text{cASO}}(W, S) = \alpha\eta(1 + \eta) \operatorname{tr} \left( W^T(\eta I + S)^{-1}W \right). \tag{14}$$

The cASO formulation in Eq. (13) and the cCMTL formulation in Eq. (11) are different in the regularization components: the respective Hessian of the regularization with respect to $W$ are different. Similar to Theorem 2.1, our analysis shows that cASO and cCMTL are equivalent.

**Theorem 3.1.** *The objectives of the cCMTL formulation in Eq. (11) and the cASO formulation in Eq. (13) are equivalent if the cluster number, $k$, in $K$-means equals to the size, $h$, of the shared low-dimensional feature space.*

*Proof.* Define the following two convex functions of $W$:

$$g_{\text{cCMTL}}(W) = \min_{M} \text{tr}\left(W(\eta I + M)^{-1}W^T\right), \quad \text{s.t. } \text{tr}(M) = k, M \preceq I, M \in \mathbb{S}^m_+, \quad (15)$$

and

$$g_{\text{cASO}}(W) = \min_{S} \text{tr}\left(W^T(\eta I + S)^{-1}W\right), \quad \text{s.t. } \text{tr}(S) = h, S \preceq I, S \in \mathbb{S}^d_+. \quad (16)$$

The cCMTL and cASO formulations can be expressed as unconstrained optimization w.r.t. $W$:

$$[\text{cCMTL}] \quad \min_{W} \mathcal{L}(W) + c \cdot g_{\text{CMTL}}(W), \qquad [\text{cASO}] \quad \min_{W} \mathcal{L}(W) + c \cdot g_{\text{ASO}}(W),$$

where $c = \alpha\eta(1 + \eta)$. Let $h = k \leq \min(d, m)$. Next, we show that for a given $W$, $g_{\text{CMTL}}(W) = g_{\text{ASO}}(W)$ holds.

Let $W = Q_1\Sigma Q_2$, $M = P_1\Lambda_1 P_1^T$, and $S = P_2\Lambda_2 P_2^T$, be the SVD of $W$, $M$, and $S$ ($M$ and $S$ are symmetric positive semi-definite), respectively, where $\Sigma = \text{diag}\{\sigma_1, \sigma_2, \ldots, \sigma_m\}$, $\Lambda_1 = \text{diag}\{\lambda_1^{(1)}, \lambda_2^{(1)}, \ldots, \lambda_m^{(1)}\}$, and $\Lambda_2 = \{\lambda_1^{(2)}, \lambda_2^{(2)}, \ldots, \lambda_m^{(2)}\}$. Let $q < k$ be the rank of $\Sigma$. It follows from the basic properties of the trace that:

$$\text{tr}\left(W(\eta I + M)^{-1}W^T\right) = \text{tr}\left((\eta I + \Lambda_1)^{-1}P_1^T Q_2\Sigma^2 Q_2^T P_1\right).$$

The problem in Eq. (15) is thus equivalent to:

$$\min_{P_1,\Lambda_1} \text{tr}\left((\eta I + \Lambda_1)^{-1}P_1^T Q_2\Sigma^2 Q_2^T P_1\right), \quad \text{s.t. } P_1 P_1^T = I, P_1^T P_1 = I, \sum_{i=1}^{d}\lambda_i^{(1)} = k. \quad (17)$$

It can be shown that the optimal $P_1^*$ is given by $P_1^* = Q_2$ and the optimal $\Lambda_1^*$ is given by solving the following simple (convex) optimization problem [13]:

$$\Lambda_1^* = \underset{\Lambda_1}{\text{argmin}} \sum_{i=1}^{q} \frac{\sigma_i^2}{\eta + \lambda_i^{(1)}}, \quad \text{s.t. } \sum_{i}\lambda_i^{(1)} = k, 0 \leq \lambda_i^{(1)} \leq 1. \quad (18)$$

It follows that $g_{\text{cCMTL}}(W) = \text{tr}\left((\eta I + \Lambda_1^*)^{-1}\Sigma^2\right)$. Similarly, we can show that $g_{\text{cASO}}(W) = \text{tr}\left((\eta I + \Lambda_2^*)^{-1}\Sigma^2\right)$, where

$$\Lambda_2^* = \underset{\Lambda_2}{\text{argmin}} \sum_{i=1}^{q} \frac{\sigma_i^2}{\eta + \lambda_i^{(2)}}, \quad \text{s.t. } \sum_{i}\lambda_i^{(2)} = h, \ 0 \leq \lambda_i^{(2)} \leq 1.$$

It is clear that when $h = k$, $\Lambda_1^* = \Lambda_2^*$ holds. Therefore, we have $g_{\text{cCMTL}}(W) = g_{\text{cASO}}(W)$. This completes the proof. $\square$

*Remark* 3.2. In the functional of cASO in Eq. (16) the variable to be optimized is $S \in \mathbb{S}^d_+$, while in the functional of cCMTL in Eq. (15) the optimization variable is $M \in \mathbb{S}^m_+$. In many practical MTL problems the data dimensionality $d$ is much larger than the task number $m$, and in such cases cCMTL is significantly more efficient in terms of both time and space. Our equivalence relationship established in Theorem 3.1 provides an (equivalent) efficient implementation of cASO especially for high-dimensional problems.

## 4 Optimization Algorithms

In this section, we propose to employ three different methods, i.e., Alternating Optimization Method (altCMTL), Accelerated Projected Gradient Method (apgCMTL), and Direct Gradient Descent Method (graCMTL), respectively, for solving the convex relaxation in Eq. (11). Note that we focus on smooth loss functions in this paper.

## 4.1 Alternating Optimization Method

The Alternating Optimization Method (altCMTL) is similar to the Block Coordinate Descent (BCD) method [22], in which the variable is optimized alternatively with the other variables fixed. The pseudo-code of altCMTL is provided in the supplemental material. Note that using similar techniques as the ones from [23], we can show that altCMTL finds the globally optimal solution to Eq. (11). The altCMTL algorithm involves the following two steps in each iteration:

**Optimization of W** For a fixed $M$, the optimal $W$ can be obtained via solving:

$$\min_W \quad \mathcal{L}(W) + c \operatorname{tr}\left(W(\eta I + M)^{-1}W^T\right). \tag{19}$$

The problem above is smooth and convex. It can be solved using gradient-type methods [22]. In the special case of a least square loss function, the problem in Eq. (19) admits an analytic solution.

**Optimization of M** For a fixed $W$, the optimal $M$ can be obtained via solving:

$$\min_M \operatorname{tr}\left(W(\eta I + M)^{-1}W^T\right), \quad \text{s.t. } \operatorname{tr}(M) = k, M \preceq I, M \in \mathbb{S}_+^m. \tag{20}$$

From Theorem 3.1, the optimal $M$ to Eq. (20) is given by $M = Q\Lambda^* Q^T$, where $\Lambda^*$ is obtained from Eq. (18). The problem in Eq. (18) can be efficiently solved using similar techniques in [17].

## 4.2 Accelerated Projected Gradient Method

The accelerated projected gradient method (APG) has been applied to solve many machine learning formulations [24]. We apply APG to solve the cCMTL formulation in Eq. (11). The algorithm is called apgCMTL. The key component of apgCMTL is to compute a proximal operator as follows:

$$\min_{W_Z, M_Z} \left\| W_Z - \hat{W}_S \right\|_F^2 + \left\| M_Z - \hat{M}_S \right\|_F^2, \quad \text{s.t.} \quad \operatorname{tr}(M_Z) = k, M_Z \preceq I, M_Z \in \mathbb{S}_+^m, \tag{21}$$

where the details about the construction of $\hat{W}_S$ and $\hat{M}_S$ can be found in [24]. The optimization problem in Eq. (21) is involved in each iteration of apgCMTL, and hence its computation is critical for the practical efficiency of apgCMTL. We show below that the optimal $W_Z$ and $M_Z$ to Eq. (21) can be computed efficiently.

**Computation of $W_z$** The optimal $W_Z$ to Eq. (21) can be obtained by solving:

$$\min_{W_Z} \left\| W_Z - \hat{W}_S \right\|_F^2. \tag{22}$$

Clearly the optimal $W_Z$ to Eq. (22) is equal to $\hat{W}_S$.

**Computation of $M_z$** The optimal $M_Z$ to Eq. (21) can be obtained by solving:

$$\min_{M_Z} \left\| M_Z - \hat{M}_S \right\|_F^2, \quad \text{s.t.} \quad \operatorname{tr}(M_Z) = k, M_Z \preceq I, M_Z \in \mathbb{S}_+^m, \tag{23}$$

where $\hat{M}_S$ is not guaranteed to be positive semidefinite. Our analysis shows that the optimization problem in Eq. (23) admits an analytical solution via solving a simple convex projection problem. The main result and the pseudo-code of apgCMTL are provided in the supplemental material.

## 4.3 Direct Gradient Descent Method

In Direct Gradient Descent Method (graCMTL) as used in [17], the cCMTL problem in Eq. (11) is reformulated as an optimization problem with one single variable $W$, given by:

$$\min_W \mathcal{L}(W) + c \cdot g_{\text{CMTL}}(W), \tag{24}$$

where $g_{\text{CMTL}}(W)$ is a functional of $W$ defined in Eq. (15).

Given the intermediate solution $W_{k-1}$ from the $(k-1)$-th iteration of graCMTL, we compute the gradient of $g_{\text{CMTL}}(W)$ and then apply the general gradient descent scheme [25] to obtain $W_k$. Note that at each iterative step in line search, we need to solve the optimization problem in the form of Eq. (20). The gradient of $g_{\text{CMTL}}(\cdot)$ at $W_{k-1}$ is given by [26, 27]: $\nabla_W g_{\text{CMTL}}(W_k) = 2(\eta I + \hat{M})^{-1}W_{k-1}^T$, where $\hat{M}$ is obtained by solving Eq. (20) at $W = W_{k-1}$. The pseudo-code of graCMTL is provided in the supplemental material.

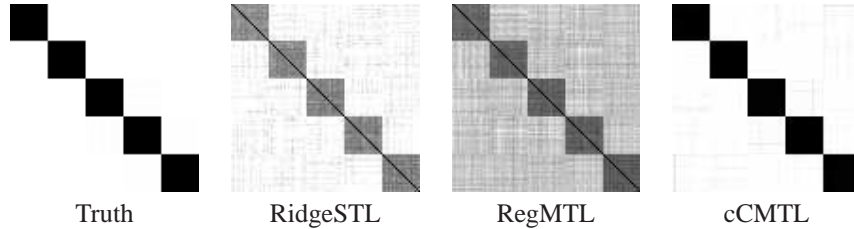

| Truth | RidgeSTL | RegMTL | cCMTL |

Figure 1: The correlation matrices of the ground truth model, and the models learnt from RidgeSTL, RegMTL, and cCMTL. Darker color indicates higher correlation. In the ground truth there are 100 tasks clustered into 5 groups. Each task has 200 dimensions. 95 training samples and 5 testing samples are used in each task. The test errors (in terms of nMSE) for RidgeSTL, RegMTL, and cCMTL are 0.8077, 0.6830, 0.0354, respectively.

## 5  Experiments

In this section, we empirically evaluate the effectiveness and the efficiency of the proposed algorithms on synthetic and real-world data sets. The normalized mean square error (nMSE) and the averaged mean square error (aMSE) are used as the performance measure [23]. Note that in this paper we have not developed new MTL formulations; instead our main focus is on the theoretical understanding of the inherent relationship between ASO and CMTL. Thus, an extensive comparative study of various MTL algorithms is out of the scope of this paper. As an illustration, in the following experiments we only compare cCMTL with two baseline techniques: ridge regression STL (RidgeSTL) and regularized MTL (RegMTL) [28].

**Simulation Study**  We apply the proposed cCMTL formulation in Eq. (11) on a synthetic data set (with a pre-defined cluster structure). We use 5-fold cross-validation to determine the regularization parameters for all methods. We construct the synthetic data set following a procedure similar to the one in [17]: the constructed synthetic data set consists of 5 clusters, where each cluster includes 20 (regression) tasks and each task is represented by a weight vector of length $d = 300$. Details of the construction is provided in the supplemental material. We apply RidgeSTL, RegMTL, and cCMTL on the constructed synthetic data. The correlation coefficient matrices of the obtained weight vectors are presented in Figure 1. From the result we can observe (1) cCMTL is able to capture the cluster structure among tasks and achieves a small test error; (2) RegMTL is better than RidgeSTL in terms of test error. It however introduces unnecessary correlation among tasks possibly due to the assumption that all tasks are related; (3) In cCMTL we also notice some 'noisy' correlation, which may because of the spectral relaxation.

Table 1: Performance comparison on the School data in terms of nMSE and aMSE. Smaller nMSE and aMSE indicate better performance. All regularization parameters are tuned using 5-fold cross validation. The mean and standard deviation are calculated based on 10 random repetitions.

| Measure | Ratio | RidgeSTL | RegMTL | cCMTL |
|---------|-------|----------|--------|-------|
| nMSE | 10% | $1.3954 \pm 0.0596$ | $1.0988 \pm 0.0178$ | $1.0850 \pm 0.0206$ |
| | 15% | $1.1370 \pm 0.0146$ | $1.0636 \pm 0.0170$ | $0.9708 \pm 0.0145$ |
| | 20% | $1.0290 \pm 0.0309$ | $1.0349 \pm 0.0091$ | $0.8864 \pm 0.0094$ |
| | 25% | $0.8649 \pm 0.0123$ | $1.0139 \pm 0.0057$ | $0.8243 \pm 0.0031$ |
| | 30% | $0.8367 \pm 0.0102$ | $1.0042 \pm 0.0066$ | $0.8006 \pm 0.0081$ |
| aMSE | 10% | $0.3664 \pm 0.0160$ | $0.2865 \pm 0.0054$ | $0.2831 \pm 0.0050$ |
| | 15% | $0.2972 \pm 0.0034$ | $0.2771 \pm 0.0045$ | $0.2525 \pm 0.0048$ |
| | 20% | $0.2717 \pm 0.0083$ | $0.2709 \pm 0.0027$ | $0.2322 \pm 0.0022$ |
| | 25% | $0.2261 \pm 0.0033$ | $0.2650 \pm 0.0027$ | $0.2154 \pm 0.0020$ |
| | 30% | $0.2196 \pm 0.0035$ | $0.2632 \pm 0.0028$ | $0.2101 \pm 0.0016$ |

**Effectiveness Comparison**  Next, we empirically evaluate the effectiveness of the cCMTL formulation in comparison with RidgeSTL and RegMTL using real world benchmark datasets including the School data[1] and the Sarcos data[2]. The regularization parameters for all algorithms are deter-

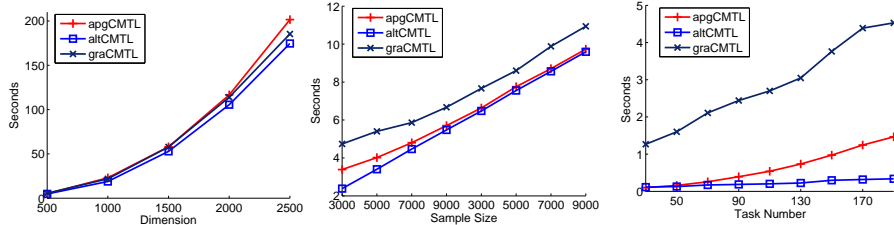

Figure 2: Sensitivity study of altCMTL, apgCMTL, graCMTL in terms of the computation cost (in seconds) with respect to feature dimensionality (left), sample size (middle), and task number (right).

mined via 5-fold cross validation; the reported experimental results are averaged over 10 random repetitions. The School data consists of the exam scores of 15362 students from 139 secondary schools, where each student is described by 27 attributes. We vary the training ratio in the set $5 \times \{1, 2, \cdots, 6\}\%$ and record the respective performance. The experimental results are presented in Table 1. We can observe that cCMTL performs the best among all settings. Experimental results on the Sarcos dataset is available in the supplemental material.

**Efficiency Comparison** We compare the efficiency of the three algorithms including altCMTL, apgCMTLand graCMTL for solving the cCMTL formulation in Eq. (11). For the following experiments, we set $\alpha = 1$, $\beta = 1$, and $k = 2$ in cCMTL. We observe a similar trend in other settings. Specifically, we study how the feature dimensionality, the sample size, and the task number affect the required computation cost (in seconds) for convergence. The experimental setup is as follows: we terminate apgCMTL when the change of objective values in two successive steps is smaller than $10^{-5}$ and record the obtained objective value; we then use such a value as the stopping criterion in graCMTL and altCMTL, that is, we stop graCMTL or altCMTL when graCMTL or altCMTL attains an objective value equal to or smaller than the one attained by apgCMTL. We use Yahoo Arts data for the first two experiments. Because in Yahoo data the task number is very small, we construct a synthetic data for the third experiment.

In the first experiment, we vary the feature dimensionality in the set $[500 : 500 : 2500]$ with the sample size fixed at 4000 and the task numbers fixed at 17. The experimental result is presented in the left plot of Figure 2. In the second experiment, we vary the sample size in the set $[3000 : 1000 : 9000]$ with the dimensionality fixed at 500 and the task number fixed at 17. The experimental result is presented in the middle plot of Figure 2. From the first two experiments, we observe that larger feature dimensionality or larger sample size will lead to higher computation cost. In the third experiment, we vary the task number in the set $[10 : 10 : 190]$ with the feature dimensionality fixed at 600 and the sample size fixed at 2000. The employed synthetic data set is constructed as follows: for each task, we generate the entries of the data matrix $X_i$ from $\mathcal{N}(0, 1)$, and generate the entries of the weight vector from $\mathcal{N}(0, 1)$, the response vector $y_i$ is computed as $y_i = X_i w_i + \xi$, where $\xi \sim \mathcal{N}(0, 0.01)$ represents the noise vector. The experimental result is presented in the right plot of Figure 2. We can observe that altCMTL is more efficient than the other two algorithms.

## 6  Conclusion

In this paper we establish the equivalence relationship between two multi-task learning techniques: alternating structure optimization (ASO) and clustered multi-task learning (CMTL). We further establish the equivalence relationship between our proposed convex relaxation of CMTL and an existing convex relaxation of ASO. In addition, we propose three algorithms for solving the convex CMTL formulation and demonstrate their effectiveness and efficiency on benchmark datasets. The proposed algorithms involve the computation of SVD. In the case of a very large task number, the SVD computation will be expensive. We seek to further improve the efficiency of the algorithms by employing approximation methods. In addition, we plan to apply the proposed algorithms to other real world applications involving multiple (clustered) tasks.

## Acknowledgments

This work was supported in part by NSF IIS-0812551, IIS-0953662, MCB-1026710, CCF-1025177, and NIH R01 LM010730.

## Footnotes

[1]http://www.cs.ucl.ac.uk/staff/A.Argyriou/code/

[2]http://gaussianprocess.org/gpml/data/

# References

[1] T. Evgeniou, M. Pontil, and O. Toubia. A convex optimization approach to modeling consumer heterogeneity in conjoint estimation. *Marketing Science*, 26(6):805–818, 2007.

[2] R.K. Ando. Applying alternating structure optimization to word sense disambiguation. In *Proceedings of the Tenth Conference on Computational Natural Language Learning*, pages 77–84, 2006.

[3] A. Torralba, K.P. Murphy, and W.T. Freeman. Sharing features: efficient boosting procedures for multi-class object detection. In *Computer Vision and Pattern Recognition, 2004, IEEE Conference on*, volume 2, pages 762–769, 2004.

[4] J. Baxter. A model of inductive bias learning. *J. Artif. Intell. Res.*, 12:149–198, 2000.

[5] R.K. Ando and T. Zhang. A framework for learning predictive structures from multiple tasks and unlabeled data. *The Journal of Machine Learning Research*, 6:1817–1853, 2005.

[6] S. Ben-David and R. Schuller. Exploiting task relatedness for multiple task learning. *Lecture notes in computer science*, pages 567–580, 2003.

[7] S. Bickel, J. Bogojeska, T. Lengauer, and T. Scheffer. Multi-task learning for hiv therapy screening. In *Proceedings of the 25th International Conference on Machine Learning*, pages 56–63. ACM, 2008.

[8] T. Evgeniou, C.A. Micchelli, and M. Pontil. Learning multiple tasks with kernel methods. *Journal of Machine Learning Research*, 6(1):615, 2006.

[9] A. Argyriou, C.A. Micchelli, M. Pontil, and Y. Ying. A spectral regularization framework for multi-task structure learning. *Advances in Neural Information Processing Systems*, 20:25–32, 2008.

[10] R. Caruana. Multitask learning. *Machine Learning*, 28(1):41–75, 1997.

[11] J. Blitzer, R. McDonald, and F. Pereira. Domain adaptation with structural correspondence learning. In *Proceedings of the 2006 Conference on EMNLP*, pages 120–128, 2006.

[12] A. Quattoni, M. Collins, and T. Darrell. Learning visual representations using images with captions. In *Computer Vision and Pattern Recognition, 2007. IEEE Conference on*, pages 1–8. IEEE, 2007.

[13] J. Chen, L. Tang, J. Liu, and J. Ye. A convex formulation for learning shared structures from multiple tasks. In *Proceedings of the 26th Annual International Conference on Machine Learning*, pages 137–144. ACM, 2009.

[14] S. Thrun and J. O'Sullivan. Clustering learning tasks and the selective cross-task transfer of knowledge. *Learning to learn*, pages 181–209, 1998.

[15] B. Bakker and T. Heskes. Task clustering and gating for bayesian multitask learning. *The Journal of Machine Learning Research*, 4:83–99, 2003.

[16] Y. Xue, X. Liao, L. Carin, and B. Krishnapuram. Multi-task learning for classification with dirichlet process priors. *The Journal of Machine Learning Research*, 8:35–63, 2007.

[17] L. Jacob, F. Bach, and J.P. Vert. Clustered multi-task learning: A convex formulation. *Arxiv preprint arXiv:0809.2085*, 2008.

[18] F. Wang, X. Wang, and T. Li. Semi-supervised multi-task learning with task regularizations. In *Data Mining, 2009. ICDM'09. Ninth IEEE International Conference on*, pages 562–568. IEEE, 2009.

[19] C. Ding and X. He. K-means clustering via principal component analysis. In *Proceedings of the twenty-first International Conference on Machine learning*, page 29. ACM, 2004.

[20] H. Zha, X. He, C. Ding, M. Gu, and H. Simon. Spectral relaxation for k-means clustering. *Advances in Neural Information Processing Systems*, 2:1057–1064, 2002.

[21] K. Fan. On a theorem of Weyl concerning eigenvalues of linear transformations I. *Proceedings of the National Academy of Sciences of the United States of America*, 35(11):652, 1949.

[22] J. Nocedal and S.J. Wright. *Numerical optimization*. Springer verlag, 1999.

[23] A. Argyriou, T. Evgeniou, and M. Pontil. Convex multi-task feature learning. *Machine Learning*, 73(3):243–272, 2008.

[24] Y. Nesterov. Gradient methods for minimizing composite objective function. *ReCALL*, 76(2007076), 2007.

[25] S.P. Boyd and L. Vandenberghe. *Convex optimization*. Cambridge University Press, 2004.

[26] J. Gauvin and F. Dubeau. Differential properties of the marginal function in mathematical programming. *Optimality and Stability in Mathematical Programming*, pages 101–119, 1982.

[27] M. Wu, B. Schölkopf, and G. Bakır. A direct method for building sparse kernel learning algorithms. *The Journal of Machine Learning Research*, 7:603–624, 2006.

[28] T. Evgeniou and M. Pontil. Regularized multi–task learning. In *Proceedings of the tenth ACM SIGKDD International Conference on Knowledge discovery and data mining*, pages 109–117. ACM, 2004.

